# Bootstrapping from Game Tree Search

**Joel Veness**
University of NSW and NICTA
Sydney, NSW, Australia 2052
`joelv@cse.unsw.edu.au`

**David Silver**
University of Alberta
Edmonton, AB Canada T6G2E8
`silver@cs.ualberta.ca`

**William Uther**
NICTA and the University of NSW
Sydney, NSW, Australia 2052
`William.Uther@nicta.com.au`

**Alan Blair**
University of NSW and NICTA
Sydney, NSW, Australia 2052
`blair@cse.unsw.edu.au`

## Abstract

In this paper we introduce a new algorithm for updating the parameters of a heuristic evaluation function, by updating the heuristic towards the values computed by an alpha-beta search. Our algorithm differs from previous approaches to learning from search, such as Samuel's checkers player and the TD-Leaf algorithm, in two key ways. First, we update all nodes in the search tree, rather than a single node. Second, we use the outcome of a deep search, instead of the outcome of a subsequent search, as the training signal for the evaluation function. We implemented our algorithm in a chess program *Meep*, using a linear heuristic function. After initialising its weight vector to small random values, *Meep* was able to learn high quality weights from self-play alone. When tested online against human opponents, *Meep* played at a master level, the best performance of any chess program with a heuristic learned entirely from self-play.

## 1 Introduction

The idea of *search bootstrapping* is to adjust the parameters of a heuristic evaluation function towards the value of a deep search. The motivation for this approach comes from the recursive nature of tree search: if the heuristic can be adjusted to match the value of a deep search of depth $D$, then a search of depth $k$ with the new heuristic would be equivalent to a search of depth $k + D$ with the old heuristic.

Deterministic, two-player games such as chess provide an ideal test-bed for search bootstrapping. The intricate tactics require a significant level of search to provide an accurate position evaluation; learning without search has produced little success in these domains. Much of the prior work in learning from search has been performed in chess or similar two-player games, allowing for clear comparisons with existing methods.

Samuel (1959) first introduced the idea of search bootstrapping in his seminal checkers player. In Samuel's work the heuristic function was updated towards the value of a minimax search in a subsequent position, after black and white had each played one move. His ideas were later extended by Baxter et al. (1998) in their chess program Knightcap. In their algorithm, TD-Leaf, the heuristic function is adjusted so that the leaf node of the principal variation produced by an alpha-beta search is moved towards the value of an alpha-beta search at a subsequent time step.

Samuel's approach and TD-Leaf suffer from three main drawbacks. First, they only update one node after each search, which discards most of the information contained in the search tree. Second, their updates are based purely on positions that have actually occurred in the game, or which lie on the computed line of best play. These positions may not be representative of the wide variety of positions that must be evaluated by a search based program; many of the positions occurring in

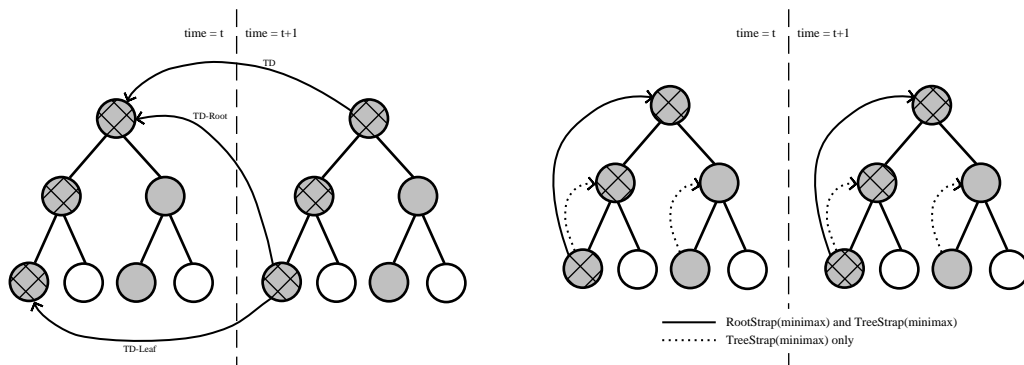

Figure 1: Left: TD, TD-Root and TD-Leaf backups. Right: RootStrap(*minimax*) and TreeStrap(*minimax*).

large search trees come from sequences of unnatural moves that deviate significantly from sensible play. Third, the target search is performed at a subsequent time-step, after a real move and response have been played. Thus, the learning target is only accurate when both the player and opponent are already strong. In practice, these methods can struggle to learn effectively from self-play alone. Work-arounds exist, such as initializing a subset of the weights to expert provided values, or by attempting to disable learning once an opponent has blundered, but these techniques are somewhat unsatisfactory if we have poor initial domain knowledge.

We introduce a new framework for bootstrapping from game tree search that differs from prior work in two key respects. First, all nodes in the search tree are updated towards the *recursive* minimax values computed by a *single* depth limited search from the root position. This makes full use of the information contained in the search tree. Furthermore, the updated positions are more representative of the types of positions that need to be accurately evaluated by a search-based player. Second, as the learning target is based on hypothetical minimax play, rather than positions that occur at subsequent time steps, our methods are less sensitive to the opponent's playing strength. We applied our algorithms to learn a heuristic function for the game of chess, starting from random initial weights and training entirely from self-play. When applied to an alpha-beta search, our chess program learnt to play at a master level against human opposition.

## 2    Background

The minimax search algorithm exhaustively computes the minimax value to some depth $D$, using a heuristic function $H_\theta(s)$ to evaluate non-terminal states at depth $D$, based on a parameter vector $\theta$. We use the notation $V_{s_0}^D(s)$ to denote the value of state $s$ in a depth $D$ minimax search from root state $s_0$. We define $T_{s_0}^D$ to be the set of states in the depth $D$ search tree from root state $s_0$. We define the *principal leaf*, $l^D(s)$, to be the leaf state of the depth $D$ principal variation from state $s$. We use the notation $\overset{\theta}{\leftarrow}$ to indicate a *backup* that updates the heuristic function towards some target value.

Temporal difference (TD) learning uses a *sample backup* $H_\theta(s_t) \overset{\theta}{\leftarrow} H_\theta(s_{t+1})$ to update the estimated value at one time-step towards the estimated value at the subsequent time-step (Sutton, 1988). Although highly successful in stochastic domains such as Backgammon (Tesauro, 1994), direct TD performs poorly in highly tactical domains. Without search or prior domain knowledge, the target value is noisy and improvements to the value function are hard to distinguish. In the game of chess, using a naive heuristic and no search, it is hard to find checkmate sequences, meaning that most games are drawn.

The quality of the target value can be significantly improved by using a *minimax backup* to update the heuristic towards the value of a minimax search. Samuel's checkers player (Samuel, 1959) introduced this idea, using an early form of bootstrapping from search that we call TD-Root. The parameters of the heuristic function, $\theta$, were adjusted towards the minimax search value at the next complete time-step (see Figure 1), $H_\theta(s_t) \overset{\theta}{\leftarrow} V_{s_{t+1}}^D(s_{t+1})$. This approach enabled Samuel's check-

ers program to achieve human amateur level play. Unfortunately, Samuel's approach was handicapped by tying his evaluation function to the material advantage, and not to the actual outcome from the position.

The TD-Leaf algorithm (Baxter et al., 1998) updates the value of a minimax search at one time-step towards the value of a minimax search at the subsequent time-step (see Figure 1). The parameters of the heuristic function are updated by gradient descent, using an update of the form $V_{s_t}^D(s_t) \xleftarrow{\theta} V_{s_{t+1}}^D(s_{t+1})$. The root value of minimax search is not differentiable in the parameters, as a small change in the heuristic value can result in the principal variation switching to a completely different path through the tree. The TD-Leaf algorithm ignores these non-differentiable boundaries by assuming that the principal variation remains unchanged, and follows the local gradient given that variation. This is equivalent to updating the heuristic function of the principal leaf, $H_\theta(l^D(s_t)) \xleftarrow{\theta} V_{s_{t+1}}^D(s_{t+1})$. The chess program Knightcap achieved master-level play when trained using TD-Leaf against a series of evenly matched human opposition, whose strength improved at a similar rate to Knightcap's. A similar algorithm was introduced contemporaneously by Beal and Smith (1997), and was used to learn the material values of chess pieces. The world champion checkers program Chinook used TD-Leaf to learn an evaluation function that compared favorably to its hand-tuned heuristic function (Schaeffer et al., 2001).

Both TD-Root and TD-Leaf are hybrid algorithms that combine a sample backup with a minimax backup, updating the current value towards the search value at a subsequent time-step. Thus the accuracy of the learning target depends both on the quality of the players, and on the quality of the search. One consequence is that these learning algorithms are not robust to variations in the training regime. In their experiments with the chess program Knightcap (Baxter et al., 1998), the authors found that it was necessary to prune training examples in which the opponent blundered or made an unpredictable move. In addition, the program was unable to learn effectively from games of self-play, and required evenly matched opposition. Perhaps most significantly, the piece values were initialised to human expert values; experiments starting from zero or random weights were unable to exceed weak amateur level. Similarly, the experiments with TD-Leaf in Chinook also fixed the important checker and king values to human expert values.

In addition, both Samuel's approach and TD-Leaf only update one node of the search tree. This does not make efficient use of the large tree of data, typically containing millions of values, that is constructed by memory enhanced minimax search variants. Furthermore, the distribution of root positions that are used to train the heuristic is very different from the distribution of positions that are evaluated during search. This can lead to inaccurate evaluation of positions that occur infrequently during real games but frequently within a large search tree; these anomalous values have a tendency to propagate up through the search tree, ultimately affecting the choice of best move at the root.

In the following section, we develop an algorithm that attempts to address these shortcomings.

## 3  Minimax Search Bootstrapping

Our first algorithm, RootStrap(*minimax*), performs a minimax search from the current position $s_t$, at every time-step $t$. The parameters are updated so as to move the heuristic value of the root node towards the minimax search value, $H_\theta(s_t) \xleftarrow{\theta} V_{s_t}^D(s_t)$. We update the parameters by stochastic gradient descent on the squared error between the heuristic value and the minimax search value. We treat the minimax search value as a constant, to ensure that we move the heuristic towards the search value, and not the other way around.

$$\delta_t = V_{s_t}^D(s_t) - H_\theta(s_t)$$
$$\Delta\theta = -\frac{\eta}{2}\nabla_\theta \delta_t^2 = \eta\delta_t\nabla_\theta H_\theta(s_t)$$

where $\eta$ is a step-size constant. RootStrap($\alpha\beta$) is equivalent to RootStrap(*minimax*), except it uses the more efficient $\alpha\beta$-search algorithm to compute $V_{s_t}^D(s_t)$.

For the remainder of this paper we consider heuristic functions that are computed by a linear combination $H_\theta(s) = \phi(s)^T\theta$, where $\phi(s)$ is a vector of features of position $s$, and $\theta$ is a parameter vector specifying the weight of each feature in the linear combination. Although simple, this form of heuristic has already proven sufficient to achieve super-human performance in the games of Chess

| Algorithm | Backup |
|---|---|
| TD | $H_\theta(s_t) \overset{\theta}{\leftarrow} H_\theta(s_{t+1})$ |
| TD-Root | $H_\theta(s_t) \overset{\theta}{\leftarrow} V^D_{s_{t+1}}(s_{t+1})$ |
| TD-Leaf | $H_\theta(l^D(s_t)) \overset{\theta}{\leftarrow} V^D_{s_{t+1}}(s_{t+1})$ |
| RootStrap(*minimax*) | $H_\theta(s_t) \overset{\theta}{\leftarrow} V^D_{s_t}(s_t)$ |
| TreeStrap(*minimax*) | $H_\theta(s) \overset{\theta}{\leftarrow} V^D_{s_t}(s), \forall s \in T^D_{s_t}$ |
| TreeStrap($\alpha\beta$) | $H_\theta(s) \overset{\theta}{\leftarrow} [b^D_{s_t}(s), a^D_{s_t}(s)], \forall s \in T^{\alpha\beta}_t$ |

Table 1: Backups for various learning algorithms.

---

**Algorithm 1** TreeStrap($minimax$)

Randomly initialise $\theta$
Initialise $t \leftarrow 1, s_1 \leftarrow$ start state
**while** $s_t$ is not terminal **do**
    $V \leftarrow \text{minimax}(s_t, H_\theta, D)$
    **for** $s \in$ search tree **do**
        $\delta \leftarrow V(s) - H_\theta(s)$
        $\Delta\theta \leftarrow \Delta\theta + \eta\delta\phi(s)$
    **end for**
    $\theta \leftarrow \theta + \Delta\theta$
    Select $a_t = \underset{a \in A}{\text{argmax}}\ V(s_t \circ a)$
    Execute move $a_t$, receive $s_{t+1}$
    $t \leftarrow t + 1$
**end while**

**Algorithm 2** DeltaFromTransTbl($s, d$)

Initialise $\Delta\theta \leftarrow \vec{0}, t \leftarrow probe(s)$
**if** $t$ is null **or** $depth(t) < d$ **then**
    **return** $\Delta\theta$
**end if**
**if** $lowerbound(t) > H_\theta(s)$ **then**
    $\Delta\theta \leftarrow \Delta\theta + \eta(lowerbound(t) - H_\theta(s))\nabla H_\theta(s)$
**end if**
**if** $upperbound(t) < H_\theta(s)$ **then**
    $\Delta\theta \leftarrow \Delta\theta + \eta(upperbound(t) - H_\theta(s))\nabla H_\theta(s)$
**end if**
**for** $s' \in succ(s)$ **do**
    $\Delta\theta \leftarrow DeltaFromTransTbl(s')$
**end for**
**return** $\Delta\theta$

---

(Campbell et al., 2002), Checkers (Schaeffer et al., 2001) and Othello (Buro, 1999). The gradient descent update for RootStrap(*minimax*) then takes the particularly simple form $\Delta\theta_t = \eta\delta_t\phi(s_t)$.

Our second algorithm, TreeStrap(*minimax*), also performs a minimax search from the current position $s_t$. However, TreeStrap(*minimax*) updates *all* interior nodes within the search tree. The parameters are updated, for each position $s$ in the tree, towards the minimax search value of $s$, $H_\theta(s) \overset{\theta}{\leftarrow} V^D_{s_t}(s), \forall s \in T^D_{s_t}$. This is again achieved by stochastic gradient descent,

$$\delta_t(s) = V^D_{s_t}(s) - H_\theta(s)$$
$$\Delta\theta = -\frac{\eta}{2}\nabla_\theta \sum_{s \in T^D_{s_t}} \delta_t(s)^2 = \eta \sum_{s \in T^D_{s_t}} \delta_t(s)\phi(s)$$

The complete algorithm for TreeStrap(*minimax*) is described in Algorithm 1.

## 4  Alpha-Beta Search Bootstrapping

The concept of minimax search bootstrapping can be extended to $\alpha\beta$-search. Unlike minimax search, alpha-beta does not compute an exact value for the majority of nodes in the search tree. Instead, the search is cut off when the value of the node is sufficiently high or low that it can no longer contribute to the principal variation. We consider a depth $D$ alpha-beta search from root position $s_0$, and denote the upper and lower bounds computed for node $s$ by $a^D_{s_0}(s)$ and $b^D_{s_0}(s)$ respectively, so that $b^D_{s_0}(s) \leq V^D_{s_0}(s) \leq a^D_{s_0}(s)$. Only one bound applies in cut off nodes: in the case of an alpha-cut we define $b^D_{s_0}(s)$ to be $-\infty$, and in the case of a beta-cut we define $a^D_{s_0}(s)$ to be $\infty$. If no cut off occurs then the bounds are exact, i.e. $a^D_{s_0}(s) = b^D_{s_0}(s) = V^D_{s_0}(s)$.

The bounded values computed by alpha-beta can be exploited by search bootstrapping, by using a one-sided loss function. If the value from the heuristic evaluation is larger than the $a$-bound of the deep search value, then it is reduced towards the $a$-bound, $H_\theta(s) \overset{\theta}{\leftarrow} a^D_{s_t}(s)$. Similarly, if the value from the heuristic evaluation is smaller than the $b$-bound of the deep search value, then it is increased

towards the $b$-bound, $H_\theta(s) \overset{\theta}{\leftarrow} b_{s_t}^D(s)$. We implement this idea by gradient descent on the sum of one-sided squared errors:

$$\delta_t^a(s) = \begin{cases} a_{s_t}^D(s) - H_\theta(s) & \text{if } H_\theta(s) > a_{s_t}^D(s) \\ 0 & \text{otherwise} \end{cases}$$

$$\delta_t^b(s) = \begin{cases} b_{s_t}^D(s) - H_\theta(s) & \text{if } H_\theta(s) < b_{s_t}^D(s) \\ 0 & \text{otherwise} \end{cases}$$

giving

$$\Delta\theta_t = \frac{\eta}{2}\nabla_\theta \sum_{s \in T_t^{\alpha\beta}} \delta_t^a(s)^2 + \delta_t^b(s)^2 = \eta \sum_{s \in T_t^{\alpha\beta}} \left( \delta_t^a(s) + \delta_t^b(s) \right) \phi(s)$$

where $T_t^{\alpha\beta}$ is the set of nodes in the alpha-beta search tree at time $t$. We call this algorithm TreeStrap($\alpha\beta$), and note that the update for each node $s$ is equivalent to the TreeStrap(*minimax*) update when no cut-off occurs.

## 4.1 Updating Parameters in TreeStrap($\alpha\beta$)

High performance $\alpha\beta$-search routines rely on transposition tables for move ordering, reducing the size of the search space, and for caching previous search results (Schaeffer, 1989). A natural way to compute $\Delta\theta$ for TreeStrap($\alpha\beta$) from a completed $\alpha\beta$-search is to recursively step through the transposition table, summing any relevant bound information. We call this procedure *DeltaFrom-TransTbl*, and give the pseudo-code for it in Algorithm 2.

*DeltaFromTransTbl* requires a standard transposition table implementation providing the following routines:

- $probe(s)$, which returns the transposition table entry associated with state $s$.
- $depth(t)$, which returns the amount of search depth used to determine the bound estimates stored in transposition table entry $t$.
- $lowerbound(t)$, which returns the lower bound stored in transposition entry $t$.
- $upperbound(t)$, which returns the upper bound stored in transposition entry $t$.

In addition, *DeltaFromTransTbl* requires a parameter $d \geq 1$, that limits updates to $\Delta\theta$ from transposition table entries based on a minimum of search depth of $d$. This can be used to control the number of positions that contribute to $\Delta\theta$ during a single update, or limit the computational overhead of the procedure.

## 4.2 The TreeStrap($\alpha\beta$) algorithm

The TreeStrap($\alpha\beta$) algorithm can be obtained by two straightforward modifications to Algorithm 1. First, the call to $minimax(s_t, H_\theta, D)$ must be replaced with a call to $\alpha\beta$-$search(s_t, H_\theta, D)$. Secondly, the inner loop computing $\Delta\theta$ is replaced by invoking $DeltaFromTransTbl(s_t)$.

# 5 Learning Chess Program

We implemented our learning algorithms in *Meep*, a modified version of the tournament chess engine *Bodo*. For our experiments, the hand-crafted evaluation function of *Bodo* was removed and replaced by a weighted linear combination of 1812 features. Given a position $s$, a feature vector $\phi(s)$ can be constructed from the 1812 numeric values of each feature. The majority of these features are binary. $\phi(s)$ is typically sparse, with approximately 100 features active in any given position. Five well-known, chess specific feature construction concepts: material, piece square tables, pawn structure, mobility and king safety were used to generate the 1812 distinct features. These features were a strict subset of the features used in *Bodo*, which are themselves simplistic compared to a typical tournament engine (Campbell et al., 2002).

The evaluation function $H_\theta(s)$ was a weighted linear combination of the features i.e. $H_\theta(s) = \phi(s)^T\theta$. All components of $\theta$ were initialised to small random numbers. Terminal positions were

evaluated as $-9999.0$, $0$ and $9999.0$ for a loss, draw and win respectively. In the search tree, mate scores were adjusted inward slightly so that shorter paths to mate were preferred when giving mate, and vice-versa. When applying the heuristic evaluation function in the search, the heuristic estimates were truncated to the interval $[-9900.0, 9900.0]$.

*Meep* contains two different modes: a tournament mode and a training mode. When in tournament mode, *Meep* uses an enhanced alpha-beta based search algorithm. Tournament mode is used for evaluating the strength of a weight configuration. In training mode however, one of two different types of game tree search algorithms are used. The first is a minimax search that stores the entire game tree in memory. This is used by the TreeStrap(*minimax*) algorithm. The second is a generic alpha-beta search implementation, that uses only three well known alpha-beta search enhancements: transposition tables, killer move tables and the history heuristic (Schaeffer, 1989). This simplified search routine was used by the TreeStrap($\alpha\beta$) and RootStrap($\alpha\beta$) algorithms. In addition, to reduce the horizon effect, checking moves were extended by one ply. During training, the transposition table was cleared before the search routine was invoked.

Simplified search algorithms were used during training to avoid complicated interactions with the more advanced heuristic search techniques (such as null move pruning) useful in tournament play. It must be stressed that during training, no heuristic or move ordering techniques dependent on knowing properties of the evaluation weights were used by the search algorithms.

Furthermore, a quiescence search (Beal, 1990) that examined all captures and check evasions was applied to leaf nodes. This was to improve the stability of the leaf node evaluations. Again, no knowledge based pruning was performed inside the quiescence search tree, which meant that the quiescence routine was considerably slower than in *Bodo*.

# 6    Experimental Results

We describe the details of our training procedures, and then proceed to explore the performance characteristics of our algorithms, RootStrap($\alpha\beta$), TreeStrap(*minimax*) and TreeStrap($\alpha\beta$) through both a large local tournament and online play. We present our results in terms of Elo ratings. This is the standard way of quantifying the strength of a chess player within a pool of players. A 300 to 500 Elo rating point difference implies a winning rate of about 85% to 95% for the higher rated player.

### 6.0.1    Training Methodology

At the start of each experiment, all weights were initialised to small random values. Games of self-play were then used to train each player. To maintain diversity during training, a small opening book was used. Once outside of the opening book, moves were selected greedily from the results of the search. Each training game was played within 1m 1s Fischer time controls. That is, both players start with a minute on the clock, and gain an additional second every time they make a move. Each training game would last roughly five minutes.

We selected the best step-size for each learning algorithm, from a series of preliminary experiments: $\alpha = 1.0 \times 10^{-5}$ for TD-Leaf and RootStrap($\alpha\beta$), $\alpha = 1.0 \times 10^{-6}$ for TreeStrap(*minimax*) and $5.0 \times 10^{-7}$ for TreeStrap($\alpha\beta$). The TreeStrap variants used a minimum search depth parameter of $d = 1$. This meant that the target values were determined by at least one ply of full-width search, plus a varying amount of quiescence search.

### 6.1    Relative Performance Evaluation

We ran a competition between many different versions of *Meep* in tournament mode, each using a heuristic function learned by one of our algorithms. In addition, a player based on randomly initialised weights was included as a reference, and arbitrarily assigned an Elo rating of 250. The best ratings achieved by each training method are displayed in Table 2.

We also measured the performance of each algorithm at intermediate stages throughout training. Figure 2 shows the performance of each learning algorithm with increasing numbers of games on a single training run. As each training game is played using the same time controls, this shows the

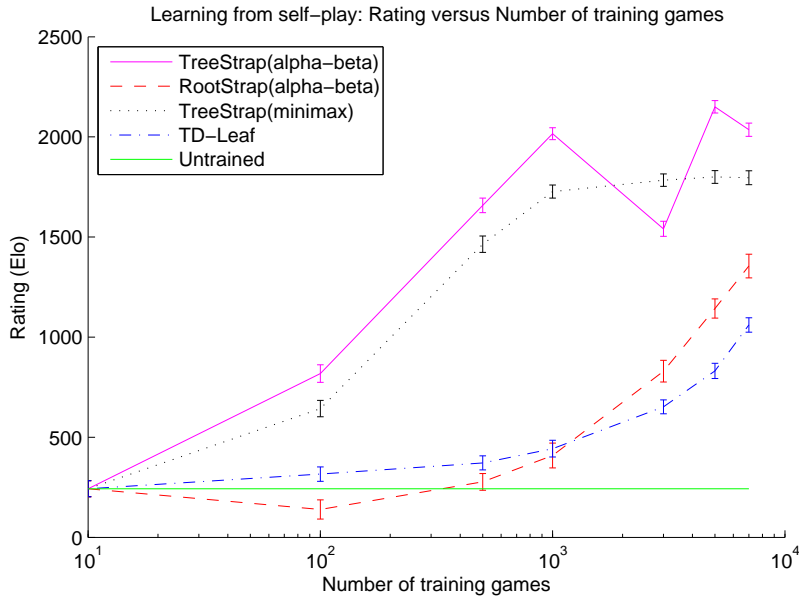

Figure 2: Performance when trained via self-play starting from random initial weights. 95% confidence intervals are marked at each data point. The $x$-axis uses a logarithmic scale.

| Algorithm | Elo |
|---|---|
| TreeStrap($\alpha\beta$) | $2157 \pm 31$ |
| TreeStrap($minimax$) | $1807 \pm 32$ |
| RootStrap($\alpha\beta$) | $1362 \pm 59$ |
| TD-Leaf | $1068 \pm 36$ |
| Untrained | $250 \pm 63$ |

Table 2: Best performance when trained by self play. 95% confidence intervals given.

performance of each learning algorithm given a fixed amount of computation. Importantly, the time used for each learning update also took away from the total thinking time.

The data shown in Table 2 and Figure 2 was generated by BayesElo, a freely available program that computes maximum likelihood Elo ratings. In each table, the estimated Elo rating is given along with a 95% confidence interval. All Elo values are calculated relative to the reference player, and should not be compared with Elo ratings of human chess players (including the results of online play, described in the next section). Approximately 16000 games were played in the tournament.

The results demonstrate that learning from many nodes in the search tree is significantly more efficient than learning from a single root node. TreeStrap(*minimax*) and TreeStrap($\alpha\beta$) learn effective weights in just a thousand training games and attain much better maximum performance within the duration of training. In addition, learning from alpha-beta search is more effective than learning from minimax search. Alpha-beta search significantly boosts the search depth, by safely pruning away subtrees that cannot affect the minimax value at the root. Although the majority of nodes now contain one-sided bounds rather than exact values, it appears that the improvements to the search depth outweigh the loss of bound information.

Our results demonstrate that the TreeStrap based algorithms can learn a good set of weights, starting from random weights, from self-play in the game of chess. Our experiences using TD-Leaf in this setting were similar to those described in (Baxter et al., 1998); within the limits of our training scheme, learning occurred, but only to the level of weak amateur play. Our results suggest that TreeStrap based methods are potentially less sensitive to initial starting conditions, and allow for speedier convergence in self play; it will be interesting to see whether similar results carry across to domains other than chess.

| Algorithm | Training Partner | Rating |
|---|---|---|
| TreeStrap($\alpha\beta$) | Self Play | 1950-2197 |
| TreeStrap($\alpha\beta$) | Shredder | 2154-2338 |

Table 3: Blitz performance at the Internet Chess Club

## 6.2 Evaluation by Internet Play

We also evaluated the performance of the heuristic function learned by TreeStrap($\alpha\beta$), by using it in *Meep* to play against predominantly human opposition at the Internet Chess Club. We evaluated two heuristic functions, the first using weights trained by self-play, and the second using weights trained against *Shredder*, a grandmaster strength commercial chess program.

The hardware used online was a 1.8Ghz Opteron, with 256Mb of RAM being used for the transposition table. Approximately 350K nodes per second were seen when using the learned evaluation function. A small opening book was used to make the engine play a variety of different opening lines. Compared to *Bodo*, the learned evaluation routine was approximately 3 times slower, even though the evaluation function contained less features. This was due to a less optimised implementation, and the heavy use of floating point arithmetic.

Approximately 1000 games were played online, using 3m 3s Fischer time controls, for each heuristic function. Although the heuristic function was fixed, the online rating fluctuates significantly over time. This is due to the high $K$ factor used by the Internet Chess Club to update Elo ratings, which is tailored to human players rather than computer engines.

The online rating of the heuristic learned by self-play corresponds to weak master level play. The heuristic learned from games against *Shredder* were roughly 150 Elo stronger, corresponding to master level performance. Like TD-Leaf, TreeStrap also benefits from a carefully chosen opponent, though the difference between self-play and ideal conditions is much less drastic. Furthermore, a total of $13.5/15$ points were scored against registered members who had achieved the title of International Master.

We expect that these results could be further improved by using more powerful hardware, a more sophisticated evaluation function, or a better opening book. Furthermore, we used a generic alpha-beta search algorithm for learning. An interesting follow-up would be to explore the interaction between our learning algorithms and the more exotic alpha-beta search enhancements.

## 7 Conclusion

Our main result is demonstrating, for the first time, an algorithm that learns to play master level Chess entirely through self play, starting from random weights. To provide insight into the nature of our algorithms, we focused on a single non-trivial domain. However, the ideas that we have introduced are rather general, and may have applications beyond deterministic two-player game tree search.

Bootstrapping from search could, in principle, be applied to many other search algorithms. Simulation-based search algorithms, such as UCT, have outperformed traditional search algorithms in a number of domains. The TreeStrap algorithm could be applied, for example, to the heuristic function that is used to initialise nodes in a UCT search tree with prior knowledge (Gelly & Silver, 2007). Alternatively, in stochastic domains the evaluation function could be updated towards the value of an expectimax search, or towards the one-sided bounds computed by a *-minimax search (Hauk et al., 2004; Veness & Blair, 2007). This approach could be viewed as a generalisation of approximate dynamic programming, in which the value function is updated from a multi-ply Bellman backup.

**Acknowledgments**

NICTA is funded by the Australian Government as represented by the Department of Broadband, Communications and the Digital Economy and the Australian Research Council through the ICT Centre of Excellence program.

# References

Baxter, J., Tridgell, A., & Weaver, L. (1998). Knightcap: a chess program that learns by combining td(lambda) with game-tree search. *Proc. 15th International Conf. on Machine Learning* (pp. 28–36). Morgan Kaufmann, San Francisco, CA.

Beal, D. F. (1990). A generalised quiescence search algorithm. *Artificial Intelligence*, *43*, 85–98.

Beal, D. F., & Smith, M. C. (1997). Learning piece values using temporal differences. *Journal of the International Computer Chess Association*.

Buro, M. (1999). From simple features to sophisticated evaluation functions. *First International Conference on Computers and Games* (pp. 126–145).

Campbell, M., Hoane, A., & Hsu, F. (2002). Deep Blue. *Artificial Intelligence*, *134*, 57–83.

Gelly, S., & Silver, D. (2007). Combining online and offline learning in UCT. *17th International Conference on Machine Learning* (pp. 273–280).

Hauk, T., Buro, M., & Schaeffer, J. (2004). Rediscovering *-minimax search. *Computers and Games* (pp. 35–50).

Samuel, A. L. (1959). Some studies in machine learning using the game of checkers. *IBM Journal of Research and Development*, *3*.

Schaeffer, J. (1989). The history heuristic and alpha-beta search enhancements in practice. *IEEE Transactions on Pattern Analysis and Machine Intelligence*, *PAMI-11*, 1203–1212.

Schaeffer, J., Hlynka, M., & Jussila, V. (2001). Temporal difference learning applied to a high performance game playing program. *IJCAI*, 529–534.

Sutton, R. (1988). Learning to predict by the method of temporal differences. *Machine Learning*, *3*, 9–44.

Tesauro, G. (1994). TD-gammon, a self-teaching backgammon program, achieves master-level play. *Neural Computation*, *6*, 215–219.

Veness, J., & Blair, A. (2007). Effective use of transposition tables in stochastic game tree search. *IEEE Symposium on Computational Intelligence and Games* (pp. 112–116).

